# Reinforcement Learning Algorithm for Partially Observable Markov Decision Problems

**Tommi Jaakkola**
tommi@psyche.mit.edu

**Satinder P. Singh**
singh@psyche.mit.edu

**Michael I. Jordan**
jordan@psyche.mit.edu

Department of Brain and Cognitive Sciences, Bld. E10
Massachusetts Institute of Technology
Cambridge, MA 02139

## Abstract

Increasing attention has been paid to reinforcement learning algorithms in recent years, partly due to successes in the theoretical analysis of their behavior in Markov environments. If the Markov assumption is removed, however, neither generally the algorithms nor the analyses continue to be usable. We propose and analyze a new learning algorithm to solve a certain class of non-Markov decision problems. Our algorithm applies to problems in which the environment is Markov, but the learner has restricted access to state information. The algorithm involves a Monte-Carlo policy evaluation combined with a policy improvement method that is similar to that of Markov decision problems and is guaranteed to converge to a local maximum. The algorithm operates in the space of stochastic policies, a space which can yield a policy that performs considerably better than any deterministic policy. Although the space of stochastic policies is continuous—even for a discrete action space—our algorithm is computationally tractable.

## 1   INTRODUCTION

Reinforcement learning provides a sound framework for credit assignment in unknown stochastic dynamic environments. For Markov environments a variety of different reinforcement learning algorithms have been devised to predict and control the environment (e.g., the TD($\lambda$) algorithm of Sutton, 1988, and the Q-learning algorithm of Watkins, 1989). Ties to the theory of dynamic programming (DP) and the theory of stochastic approximation have been exploited, providing tools that have allowed these algorithms to be analyzed theoretically (Dayan, 1992; Tsitsiklis, 1994; Jaakkola, Jordan, & Singh, 1994; Watkins & Dayan, 1992).

Although current reinforcement learning algorithms are based on the assumption that the learning problem can be cast as Markov decision problem (MDP), many practical problems resist being treated as an MDP. Unfortunately, if the Markov assumption is removed examples can be found where current algorithms cease to perform well (Singh, Jaakkola, & Jordan, 1994). Moreover, the theoretical analyses rely heavily on the Markov assumption.

The non-Markov nature of the environment can arise in many ways. The most direct extension of MDP's is to deprive the learner of perfect information about the state of the environment. Much as in the case of Hidden Markov Models (HMM's), the underlying environment is assumed to be Markov, but the data do not appear to be Markovian to the learner. This extension not only allows for a tractable theoretical analysis, but is also appealing for practical purposes. The decision problems we consider here are of this type.

The analog of the HMM for control problems is the partially observable Markov decision process (POMDP; see e.g., Monahan, 1982). Unlike HMM's, however, there is no known computationally tractable procedure for POMDP's. The problem is that once the state estimates have been obtained, DP must be performed in the continuous space of probabilities of state occupancies, and this DP process is computationally infeasible except for small state spaces. In this paper we describe an alternative approach for POMDP's that avoids the state estimation problem and works directly in the space of (stochastic) control policies. (See Singh, et al., 1994, for additional material on stochastic policies.)

## 2   PARTIAL OBSERVABILITY

A Markov decision problem can be generalized to a POMDP by restricting the state information available to the learner. Accordingly, we define the learning problem as follows. There is an underlying MDP with states $\mathcal{S} = \{s_1, s_2, \ldots, s_N\}$ and transition probability $p_{ss'}^a$, the probability of jumping from state $s$ to state $s'$ when action $a$ is taken in state $s$. For every state and every action a (random) reward is provided to the learner. In the POMDP setting, the learner is not allowed to observe the state directly but only via messages containing information about the state. At each time step $t$ an observable message $m_t$ is drawn from a finite set of messages according to an unknown probability distribution $P(m|s_t)$ [1]. We assume that the learner does

not possess any prior information about the underlying MDP beyond the number of messages and actions. The goal for the learner is to come up with a policy—a mapping from messages to actions—that gives the highest expected reward.

As discussed in Singh et al. (1994), stochastic policies can yield considerably higher expected rewards than deterministic policies in the case of POMDP's. To make this statement precise requires an appropriate technical definition of "expected reward," because in general it is impossible to find a policy, stochastic or not, that maximizes the expected reward for each observable message separately. We take the time-average reward as a measure of performance, that is, the total accrued reward per number of steps taken (Bertsekas, 1987; Schwartz, 1993). This approach requires the assumption that every state of the underlying controllable Markov chain is reachable.

In this paper we focus on a *direct* approach to solving the learning problem. Direct approaches are to be compared to *indirect* approaches, in which the learner first identifies the parameters of the underlying MDP, and then utilizes DP to obtain the policy. As we noted earlier, indirect approaches lead to computationally intractable algorithms. Our approach can be viewed as providing a generalization of the direct approach to MDP's to the case of POMDP's.

## 3  A MONTE-CARLO POLICY EVALUATION

Advantages of Monte-Carlo methods for policy evaluation in MDP's have been reviewed recently (Barto and Duff, 1994). Here we present a method for calculating the value of a stochastic policy that has the flavor of a Monte-Carlo algorithm. To motivate such an approach let us first consider a simple case where the average reward is known and generalize the well-defined MDP value function to the POMDP setting. In the Markov case the value function can be written as (cf. Bertsekas, 1987):

$$V(s) = \lim_{N \to \infty} \sum_{t=1}^{N} E\{R(s_t, u_t) - R | s_1 = s\} \tag{1}$$

where $s_t$ and $a_t$ refer to the state and the action taken at the $t^{th}$ step respectively. This form generalizes easily to the level of messages by taking an additional expectation:

$$V(m) = E\{V(s) | s \to m\} \tag{2}$$

where $s \to m$ refers to all the instances where $m$ is observed in $s$ and $E\{\cdot | s \to m\}$ is a Monte-Carlo expectation. This generalization yields a POMDP value function given by

$$V(m) = \sum_{s \in m} P(s|m) V(s) \tag{3}$$

in which $P(s|m)$ define the limit occupancy probabilities over the underlying states for each message $m$. As is seen in the next section value functions of this type can be used to refine the currently followed control policy to yield a higher average reward.

Let us now consider how the generalized value functions can be computed based on the observations. We propose a recursive Monte-Carlo algorithm to effectively compute the averages involved in the definition of the value function. In the simple

case when the average payoff is known this algorithm is given by

$$\beta_t(m) = (1 - \frac{\chi_t(m)}{K_t(m)})\gamma_t\beta_{t-1}(m) + \frac{\chi_t(m)}{K_t(m)} \tag{4}$$

$$V_t(m) = (1 - \frac{\chi_t(m)}{K_t(m)})V_{t-1}(m) + \beta_t(m)[R(s_t, a_t) - R] \tag{5}$$

where $\chi_t(m)$ is the indicator function for message $m$, $K_t(m)$ is the number of times $m$ has occurred, and $\gamma_t$ is a discount factor converging to one in the limit. This algorithm can be viewed as recursive averaging of (discounted) sample sequences of different lengths each of which has been started at a different occurrence of message $m$. This can be seen by unfolding the recursion, yielding an explicit expression for $V_t(m)$. To this end, let $t_k$ denote the time step corresponding to the $k^{th}$ occurrence of message $m$ and for clarity let $R_t = R(s_t, u_t) - R$ for every $t$. Using these the recursion yields:

$$V_t(m) = \frac{1}{K_t(m)}[\ R_{t_1} + \Gamma_{1,1}\,R_{t_1+1} + \ldots + \Gamma_{1,t-t_1}\,R_t$$
$$\ldots$$
$$+ R_{t_k} + \Gamma_{k,1}\,R_{t_k+1} + \ldots + \Gamma_{k,t-t_k}\,R_t] \tag{6}$$

where we have for simplicity used $\Gamma_{k,T}$ to indicate the discounting at the $T^{th}$ step in the $k^{th}$ sequence. Comparing the above expression to equation 1 indicates that the discount factor has to converge to one in the limit since the averages in $V(s)$ or $V(m)$ involve no discounting.

To address the question of convergence of this algorithm let us first assume a constant discounting (that is, $\gamma_t = \gamma < 1$). In this case, the algorithm produces at best an approximation to the value function. For large $K(m)$ the convergence rate by which this approximate solution is found can be characterized in terms of the bias and variance. This gives $Bias\{V(m)\} \propto (1 - \bar{\gamma})^{-1}/K(m)$ and $Var\{V(m)\} \propto (1 - \bar{\gamma})^{-2}/K(m)$ where $\bar{\gamma} = E\{\gamma^{t_k - t_{k-1}}\}$ is the expected effective discounting between observations. Now, in order to find the correct value function we need an appropriate way of letting $\gamma_t \to 1$ in the limit. However, not all such schedules lead to convergent algorithms; setting $\gamma_t = 1$ for all $t$, for example, would not. By making use of the above bounds a feasible schedule guaranteeing a vanishing bias and variance can be found. For instance, since $\gamma > \bar{\gamma}$ we can choose $\gamma_{k(m)} = 1 - K(m)^{1/4}$. Much faster schedules are possible to obtain by estimating $\bar{\gamma}$.

Let us now revise the algorithm to take into account the fact that the learner in fact has no prior knowledge of the average reward. In this case the true average reward appearing in the above algorithm needs to be replaced with an incrementally updated estimate $R_{t-1}$. To improve the effect this changing estimate has on the values we transform the value function whenever the estimate is updated. This transformation is given by

$$C_t(m) = (1 - \frac{\chi_t(m)}{K_t(m)})C_{t-1}(m) + \beta_t(m) \tag{7}$$

$$V_t(m) \to V_t(m) - C_t(m)(R_t - R_{t-1}) \tag{8}$$

and, as a result, the new values are as if they had been computed using the current estimate of the average reward.

To carry these results to the control setting and assign a figure of merit to stochastic policies we need a quantity related to the actions for each observed message. As in the case of MDP's, this is readily achieved by replacing $m$ in the algorithm just described by $(m, a)$. In terms of equation 6, for example, this means that the sequences started from $m$ are classified according to the actions taken when $m$ is observed. The above analysis goes through when $m$ is replaced by $(m, a)$, yielding "Q-values" on the level of messages:

$$Q^\pi(m, a) = \sum_s P^\pi(s|m) Q^\pi(s, a) \qquad (9)$$

In the next section we show how these values can be used to search efficiently for a better policy.

## 4  POLICY IMPROVEMENT THEOREM

Here we present a policy improvement theorem that enables the learner to search efficiently for a better policy in the continuous policy space using the "Q-values" $Q(m, a)$ described in the previous section. The theorem allows the policy refinement to be done in a way that is similar to policy improvement in a MDP setting.

**Theorem 1** *Let the current stochastic policy $\pi(a|m)$ lead to Q-values $Q^\pi(m, a)$ on the level of messages. For any policy $\pi^1(a|m)$ define*

$$J^{\pi^1}(m) = \sum_a \pi^1(a|m)[Q^\pi(m, a) - V^\pi(m)]$$

*The change in the average reward resulting from changing the current policy according to $\pi(a|m) \to (1 - \epsilon)\pi(a|m) + \epsilon\pi^1(a|m)$ is given by*

$$\Delta R^\pi = \epsilon \sum_m P^\pi(m) J^{\pi^1}(m) + O(\epsilon^2)$$

*where $P^\pi(m)$ are the occupancy probabilities for messages associated with the current policy.*

The proof is given in Appendix. In terms of policy improvement the theorem can be interpreted as follows. Choose the policy $\pi^1(a|m)$ such that

$$J^{\pi^1}(m) = \max_a[Q^\pi(m, a) - V^\pi(m)] \qquad (10)$$

If now $J^{\pi^1}(m) > 0$ for some $m$ then we can change the current policy towards $\pi^1$ and expect an increase in the average reward as shown by the theorem. The $\epsilon$ factor suggests local changes in the policy space and the policy can be refined until $\max_{\pi^1} J^{\pi^1}(m) = 0$ for all $m$ which constitutes a local maximum for this policy improvement method. Note that the new direction $\pi^1(a|m)$ in the policy space can be chosen separately for each $m$.

## 5  THE ALGORITHM

Based on the theoretical analysis presented above we can construct an algorithm that performs well in a POMDP setting. The algorithm is composed of two parts: First,

$Q(m, a)$ values—analogous to the Q-values in MDP—are calculated via a Monte-Carlo approach. This is followed by a policy improvement step which is achieved by increasing the probability of taking the best action as defined by $Q(m, a)$. The new policy is guaranteed to yield a higher average reward (see Theorem 1) as long as for some $m$

$$\max_a [Q(m, a) - V(m)] > 0 \qquad (11)$$

This condition being false constitutes a local maximum for the algorithm. Examples illustrating that this indeed is a local maximum can be found fairly easily.

In practice, it is not feasible to wait for the Monte-Carlo policy evaluation to converge but to try to improve the policy before the convergence. The policy can be refined concurrently with the Monte-Carlo method according to

$$\pi(a|m_n) \rightarrow \pi(a|m_n) + \epsilon[Q_n(m_n, a) - V_n(m_n)] \qquad (12)$$

with normalization. Other asynchronous or synchronous on-online updating schemes can also be used. Note that if $Q_n(m, a) = Q(m, a)$ then this change would be statistically equivalent to that of the batch version with the concomitant guarantees of giving a higher average reward.

# 6   CONCLUSIONS

In this paper we have proposed and theoretically analyzed an algorithm that solves a reinforcement learning problem in a POMDP setting, where the learner has restricted access to the state of the environment. As the underlying MDP is not known the problem appears to the learner to have a non-Markov nature. The average reward was chosen as the figure of merit for the learning problem and stochastic policies were used to provide higher average rewards than can be achieved with deterministic policies. This extension from MDP's to POMDP's greatly increases the domain of potential applications of reinforcement learning methods.

The simplicity of the algorithm stems partly from a Monte-Carlo approach to obtaining action-dependent values for each message. These new "Q-values" were shown to give rise to a simple policy improvement result that enables the learner to gradually improve the policy in the continuous space of probabilistic policies.

The batch version of the algorithm was shown to converge to a local maximum. We also proposed an on-line version of the algorithm in which the policy is changed concurrently with the calculation of the "Q-values." The policy improvement of the on-line version resembles that of learning automata.

**APPENDIX**

Let us denote the policy after the change by $\pi^\epsilon$. Assume first that we have access to $Q^\pi(s, a)$, the Q-values for the underlying MDP, and to $P^{\pi^\epsilon}(s|m)$, the occupancy probabilities after the policy refinement. Define

$$J(m, \pi^\epsilon, \pi^\epsilon, \pi) = \sum_a \pi^\epsilon(a|m) \sum_{s \in m} P^{\pi^\epsilon}(s|m)[Q^\pi(s, a) - V^\pi(s)] \qquad (13)$$

where we have used the notation that the policies on the left hand side correspond to the policies on the right respectively. To show how the average reward depends

on this quantity we need to make use of the following facts. The Q-values for the underlying MDP satisfy (Bellman's equation)

$$Q^\pi(s,a) = R(s,a) - R^\pi + \sum_{s'} p^a_{ss'} V^\pi(s') \tag{14}$$

In addition, $\sum_a \pi(a|m)Q^\pi(s,a) = V^\pi(s)$, implying that $J(m, \pi^\epsilon, \pi^\epsilon, \pi^\epsilon) = 0$ (see eq. 13). These facts allow us to write

$$
\begin{aligned}
J(m, \pi^\epsilon, \pi^\epsilon, \pi) &= J(m, \pi^\epsilon, \pi^\epsilon, \pi) - J(m, \pi^\epsilon, \pi^\epsilon, \pi^\epsilon) \\
&= \sum_a \pi^\epsilon(a|m) \sum_s P^{\pi^\epsilon}(s|m)[Q^\pi(s,a) - V^\pi(s) - Q^{\pi^\epsilon}(s,a) + V^{\pi^\epsilon}(s)] \\
&= R^{\pi^\epsilon} - R^\pi + \sum_s P^{\pi^\epsilon}(s|m) \sum_{s'} p^{\pi^\epsilon}_{ss'} [V^\pi(s') - V^{\pi^\epsilon}(s')] \\
&\quad - \sum_s P^{\pi^\epsilon}(s|m)[V^\pi(s) - V^{\pi^\epsilon}(s)]
\end{aligned}
\tag{15}
$$

By weighting this result for each class by $P^{\pi^\epsilon}(m)$ and summing over the messages the probability weightings for the last two terms become equal and the terms cancel. This procedure gives us

$$R^{\pi^\epsilon} - R^\pi = \sum_m P^{\pi^\epsilon}(m) J(m, \pi^\epsilon, \pi^\epsilon, \pi) \tag{16}$$

This result does not allow the learner to assess the effect of the policy refinement on the average reward since the $J()$ term contains information not available to the learner. However, making use of the fact that the policy has been changed only slightly this problem can be avoided.

As $\pi^\epsilon$ is a policy satisfying $\max_{ma} |\pi^\epsilon(a|m) - \pi(a|m)| \leq \epsilon$, it can then be shown that there exists a constant $C$ such that the maximum change in $P(s|m)$, $P(s)$, $P(m)$ is bounded by $C\epsilon$. Using these bounds and indicating the difference between $\pi^\epsilon$ and $\pi$ dependent quantities by $\Delta$ we get

$$
\begin{aligned}
&\sum_a [\pi(a|m) + \Delta\pi(a|m)] \sum_s [P^\pi(s|m) + \Delta P^\pi(s|m)][Q^\pi(s,a) - V^\pi(s)] \\
&= \sum_a \Delta\pi(a|m) \sum_{s \in m} P^\pi(s|m)[Q^\pi(s,a) - V^\pi(s)] + \\
&\quad + \sum_a \Delta\pi(a|m) \sum_s \Delta P^\pi(s|m)[Q^\pi(s,a) - V^\pi(s)] \\
&= \epsilon \sum_a \pi^1(a|m) \sum_s P^\pi(s|m)[Q^\pi(s,a) - V^\pi(s)] + O(\epsilon^2)
\end{aligned}
\tag{17}
$$

where the second equality follows from $\sum_a \pi(a|m)[Q^\pi(s,a) - V^\pi(s)] = 0$ and the third from the bounds stated earlier.

The equation characterizing the change in the average reward due to the policy change (eq. 16) can be now rewritten as follows:

$$R^{\pi^\epsilon} - R^\pi = \sum_m P^{\pi^\epsilon}(m) J(m, \pi^\epsilon, \pi, \pi) + O(\epsilon^2)$$

$$= \sum_m P^\pi(m) \sum_a \pi^\epsilon(a|m)[Q^\pi(m,a) - V^\pi(m)] + O(\epsilon^2) \qquad (18)$$

where the bounds (see above) have been used for $P^{\pi^\epsilon}(m) - P^\pi(m)$. This completes the proof. $\qquad\qquad\qquad\qquad\qquad\qquad\qquad\qquad\qquad\qquad\qquad\qquad\qquad$ □

## Acknowledgments

The authors thank Rich Sutton for pointing out errors at early stages of this work. This project was supported in part by a grant from the McDonnell-Pew Foundation, by a grant from ATR Human Information Processing Research Laboratories, by a grant from Siemens Corporation and by grant N00014-94-1-0777 from the Office of Naval Research. Michael I. Jordan is a NSF Presidential Young Investigator.

## Footnotes

[1] For simplicity we assume that this distribution depends only on the current state. The analyses go through also with distributions dependent on the past states and actions

## References

Barto, A., and Duff, M. (1994). Monte-Carlo matrix inversion and reinforcement learning. In *Advances of Neural Information Processing Systems 6*, San Mateo, CA, 1994. Morgan Kaufmann.

Bertsekas, D. P. (1987). *Dynamic Programming: Deterministic and Stochastic Models*. Englewood Cliffs, NJ: Prentice-Hall.

Dayan, P. (1992). The convergence of TD($\lambda$) for general $\lambda$. *Machine Learning, 8,* 341-362.

Jaakkola, T., Jordan M. I., and Singh, S. P. (1994). On the convergence of stochastic iterative Dynamic Programming algorithms. *Neural Computation 6,* 1185-1201.

Monahan, G. (1982). A survey of partially observable Markov decision processes. *Management Science, 28,* 1-16.

Singh, S. P., Jaakkola, T., Jordan, M. I. (1994). Learning without state estimation in partially observable environments. In *Proceedings of the Eleventh Machine Learning Conference.*

Sutton, R. S. (1988). Learning to predict by the methods of temporal differences. *Machine Learning, 3,* 9-44.

Schwartz, A. (1993). A reinforcement learning method for maximizing undiscounted rewards. In *Proceedings of the Tenth Machine Learning Conference.*

Tsitsiklis J. N. (1994). Asynchronous stochastic approximation and Q-learning. *Machine Learning 16,* 185-202.

Watkins, C.J.C.H. (1989). *Learning from delayed rewards.* PhD Thesis, University of Cambridge, England.

Watkins, C.J.C.H, & Dayan, P. (1992). Q-learning. *Machine Learning, 8,* 279-292.